# 2D Observers for Human 3D Object Recognition?

Zili Liu
NEC Research Institute

Daniel Kersten
University of Minnesota

## Abstract

Converging evidence has shown that human object recognition depends on familiarity with the images of an object. Further, the greater the similarity between objects, the stronger is the dependence on object appearance, and the more important two-dimensional (2D) image information becomes. These findings, however, do not rule out the use of 3D structural information in recognition, and the degree to which 3D information is used in visual memory is an important issue. Liu, Knill, & Kersten (1995) showed that any model that is restricted to rotations in the image plane of independent 2D templates could not account for human performance in discriminating novel object views. We now present results from models of generalized radial basis functions (GRBF), 2D nearest neighbor matching that allows 2D affine transformations, and a Bayesian statistical estimator that integrates over all possible 2D affine transformations. The performance of the human observers relative to each of the models is better for the novel views than for the familiar template views, suggesting that humans generalize better to novel views from template views. The Bayesian estimator yields the optimal performance with 2D affine transformations and independent 2D templates. Therefore, models of 2D affine matching operations with independent 2D templates are unlikely to account for human recognition performance.

## 1   Introduction

Object recognition is one of the most important functions in human vision. To understand human object recognition, it is essential to understand how objects are represented in human visual memory. A central component in object recognition is the matching of the stored object representation with that derived from the image input. But the nature of the object representation has to be inferred from recognition performance, by taking into account the contribution from the image information. When evaluating human performance, how can one separate the con-

tributions to performance of the image information from the representation? Ideal observer analysis provides a precise computational tool to answer this question. An ideal observer's recognition performance is restricted only by the available image information and is otherwise optimal, in the sense of statistical decision theory, irrespective of how the model is implemented. A comparison of human to ideal performance (often in terms of efficiency) serves to normalize performance with respect to the image information for the task. We consider the problem of viewpoint dependence in human recognition.

A recent debate in human object recognition has focused on the dependence of recognition performance on viewpoint [1, 6]. Depending on the experimental conditions, an observer's ability to recognize a familiar object from novel viewpoints is impaired to varying degrees. A central assumption in the debate is the equivalence in viewpoint dependence and recognition performance. In other words, the assumption is that viewpoint dependent performance implies a viewpoint dependent representation, and that viewpoint independent performance implies a viewpoint independent representation. However, given that any recognition performance depends on the input image information, which is necessarily viewpoint dependent, the viewpoint dependence of the performance is neither necessary nor sufficient for the viewpoint dependence of the representation. Image information has to be factored out first, and the ideal observer provides the means to do this.

The second aspect of an ideal observer is that it is implementation free. Consider the GRBF model [5], as compared with human object recognition (see below). The model stores a number of 2D templates $\{T_i\}$ of a 3D object $O$, and recognizes or rejects a stimulus image $S$ by the following similarity measure $\Sigma_i c_i \exp\left(\|T_i - S\|^2 / 2\sigma^2\right)$, where $c_i$ and $\sigma$ are constants. The model's performance as a function of viewpoint parallels that of human observers. This observation has led to the conclusion that the human visual system may indeed, as does the model, use 2D stored views with GRBF interpolation to recognize 3D objects [2]. Such a conclusion, however, overlooks implementational constraints in the model, because the model's performance also depends on its implementations. Conceivably, a model with some 3D information of the objects can also mimic human performance, so long as it is appropriately implemented. There are typically too many possible models that can produce the same pattern of results.

In contrast, an ideal observer computes the optimal performance that is only limited by the stimulus information and the task. We can define constrained ideals that are also limited by explicitly specified assumptions (e.g., a class of matching operations). Such a model observer therefore yields the best possible performance among the class of models with the same stimulus input and assumptions. In this paper, we are particularly interested in constrained ideal observers that are restricted in functionally significant aspects (e.g., a 2D ideal observer that stores independent 2D templates and has access only to 2D affine transformations). The key idea is that a constrained ideal observer is the best in its class. So if humans outperform this ideal observer, they must have used more than what is available to the ideal. The conclusion that follows is strong: not only does the constrained ideal fail to account for human performance, but the whole class of its implementations are also falsified.

A crucial question in object recognition is the extent to which human observers model the geometric variation in images due to the projection of a 3D object onto a 2D image. At one extreme, we have shown that any model that compares the image to independent views (even if we allow for 2D rigid transformations of the input image) is insufficient to account for human performance . At the other extreme, it is unlikely that variation is modeled in terms of rigid transformation of a 3D object

template in memory. A possible intermediate solution is to match the input image to stored views, subject to 2D affine deformations. This is reasonable because 2D affine transformations approximate 3D variation over a limited range of viewpoint change.

In this study, we test whether any model limited to the independent comparison of 2D views, but with 2D affine flexibility, is sufficient to account for viewpoint dependence in human recognition. In the following section, we first define our experimental task, in which the computational models yield the provably best possible performance under their specified conditions. We then review the 2D ideal observer and GRBF model derived in [4], and the 2D affine nearest neighbor model in [8]. Our principal theoretical result is a closed-form solution of a Bayesian 2D affine ideal observer. We then compare human performance with the 2D affine ideal model, as well as the other three models. In particular, if humans can classify novel views of an object better than the 2D affine ideal, then our human observers must have used more information than that embodied by that ideal.

## 2   The observers

Let us first define the task. An observer looks at the 2D images of a 3D wire frame object from a number of viewpoints. These images will be called templates $\{\mathbf{T}_i\}$. Then two distorted copies of the original 3D object are displayed. They are obtained by adding 3D Gaussian positional noise (i.i.d.) to the vertices of the original object. One distorted object is called the target, whose Gaussian noise has a constant variance. The other is the distractor, whose noise has a larger variance that can be adjusted to achieve a criterion level of performance. The two objects are displayed from the same viewpoint in parallel projection, which is either from one of the template views, or a novel view due to 3D rotation. The task is to choose the one that is more similar to the original object. The observer's performance is measured by the variance (threshold) that gives rise to 75% correct performance. The optimal strategy is to choose the stimulus $\mathbf{S}$ with a larger probability $p(\mathbf{O}|\mathbf{S})$. From Bayes' rule, this is to choose the larger of $p(\mathbf{S}|\mathbf{O})$.

Assume that the models are restricted to 2D transformations of the image, and cannot reconstruct the 3D structure of the object from its independent templates $\{\mathbf{T}_i\}$. Assume also that the prior probability $p(\mathbf{T}_i)$ is constant. Let us represent $\mathbf{S}$ and $\mathbf{T}_i$ by their $(x, y)$ vertex coordinates: $(\ \mathbf{X}\quad \mathbf{Y}\ )^T$, where $\mathbf{X} = (x^1, x^2, \ldots, x^n)$, $\mathbf{Y} = (y^1, y^2, \ldots, y^n)$. We assume that the correspondence between $\mathbf{S}$ and $\mathbf{T}_i$ is solved up to a reflection ambiguity, which is equivalent to an additional template: $\mathbf{T}_i^r = (\ \mathbf{X}^r\quad \mathbf{Y}^r\ )^T$, where $\mathbf{X}^r = (x^n, \ldots, x^2, x^1)$, $\mathbf{Y}^r = (y^n, \ldots, y^2, y^1)$. We still denote the template set as $\{\mathbf{T}_i\}$. Therefore,

$$p(\mathbf{S}|\mathbf{O}) = \Sigma p(\mathbf{S}|\mathbf{T}_i)p(\mathbf{T}_i). \tag{1}$$

In what follows, we will compute $p(\mathbf{S}|\mathbf{T}_i)p(\mathbf{T}_i)$, with the assumption that $\mathbf{S} = \mathcal{F}(\mathbf{T}_i) + \mathbf{N}(\mathbf{0}, \sigma \mathbf{I}_{2n})$, where $\mathbf{N}$ is the Gaussian distribution, $\mathbf{I}_{2n}$ the $2n \times 2n$ identity matrix, and $\mathcal{F}$ a 2D transformation. For the 2D ideal observer, $\mathcal{F}$ is a rigid 2D rotation. For the GRBF model, $\mathcal{F}$ assigns a linear coefficient to each template $\mathbf{T}_i$, in addition to a 2D rotation. For the 2D affine nearest neighbor model, $\mathcal{F}$ represents the 2D affine transformation that minimizes $\|\mathbf{S} - \mathbf{T}_i\|^2$, after $\mathbf{S}$ and $\mathbf{T}_i$ are normalized in size. For the 2D affine ideal observer, $\mathcal{F}$ represents all possible 2D affine transformations applicable to $\mathbf{T}_i$.

## 2.1   The 2D ideal observer

The templates are the original 2D images, their mirror reflections, and 2D rotations (in angle $\phi$) in the image plane. Assume that the stimulus S is generated by adding Gaussian noise to a template, the probability $p(\mathbf{S}|\mathbf{O})$ is an integration over all templates and their reflections and rotations. The detailed derivation for the 2D ideal and the GRBF model can be found in [4].

$$\Sigma p(\mathbf{S}|\mathbf{T}_i)p(\mathbf{T}_i) \propto \Sigma \int d\phi \exp\left(-\|\mathbf{S} - \mathbf{T}_i(\phi)\|^2/2\sigma^2\right). \tag{2}$$

## 2.2   The GRBF model

The model has the same template set as the 2D ideal observer does. Its training requires that $\Sigma_i \int_0^{2\pi} d\phi c_i(\phi)N(\|\mathbf{T}_j - \mathbf{T}_i(\phi)\|, \sigma) = 1$, $j = 1, 2, \ldots$, with which $\{c_i\}$ can be obtained optimally using singular value decomposition. When a pair of new stimuli $\{\mathbf{S}\}$ are presented, the optimal decision is to choose the one that is closer to the learned prototype, in other words, the one with a smaller value of

$$\|1 - \Sigma \int_0^{2\pi} d\phi c_i(\phi) \exp\left(-\frac{\|\mathbf{S} - \mathbf{T}_i(\phi)\|^2}{2\sigma^2}\right)\|. \tag{3}$$

## 2.3   The 2D affine nearest neighbor model

It has been proved in [8] that the smallest Euclidean distance $D(\mathbf{S}, \mathbf{T})$ between $\mathbf{S}$ and $\mathbf{T}$ is, when $\mathbf{T}$ is allowed a 2D affine transformation, $\mathbf{S} \to \mathbf{S}/\|\mathbf{S}\|$, $\mathbf{T} \to \mathbf{T}/\|\mathbf{T}\|$,

$$D^2(\mathbf{S}, \mathbf{T}) = 1 - tr(\mathbf{S}^+\mathbf{S} \cdot \mathbf{T}^T\mathbf{T})/\|\mathbf{T}\|^2, \tag{4}$$

where $tr$ strands for $trace$, and $\mathbf{S}^+ = \mathbf{S}^T(\mathbf{S}\mathbf{S}^T)^{-1}$. The optimal strategy, therefore, is to choose the $\mathbf{S}$ that gives rise to the larger of $\Sigma \exp\left(-D^2(\mathbf{S}, \mathbf{T}_i)/2\sigma^2\right)$, or the smaller of $\Sigma D^2(\mathbf{S}, \mathbf{T}_i)$. (Since no probability is defined in this model, both measures will be used and the results from the better one will be reported.)

## 2.4   The 2D affine ideal observer

We now calculate the Bayesian probability by assuming that the prior probability distribution of the 2D affine transformation, which is applied to the template $\mathbf{T}_i$, $\mathbf{AT} + \mathbf{T_r} = \begin{pmatrix} a & b \\ c & d \end{pmatrix}\mathbf{T}_i + \begin{pmatrix} t_x & \cdots & t_x \\ t_y & \cdots & t_y \end{pmatrix}$, obeys a Gaussian distribution $\mathbf{N}(\mathbf{X}_0, \gamma\mathbf{I}_6)$, where $\mathbf{X}_0$ is the identity transformation $\mathbf{X}_0^T = (a, b, c, d, t_x, t_y) = (1, 0, 0, 1, 0, 0)$. We have

$$\Sigma p(\mathbf{S}|\mathbf{T}_i) = \Sigma \int_{-\infty}^{\infty} d\mathbf{X} \exp\left(-\|\mathbf{AT}_i + \mathbf{T_r} - \mathbf{S}\|^2/2\sigma^2\right) \tag{5}$$

$$= \Sigma C(n, \sigma, \gamma)det^{-1}\left(\mathbf{Q}_i'\right) \exp\left(tr\left(\mathbf{K}_i^T\mathbf{Q}_i(\mathbf{Q}_i')^{-1}\mathbf{Q}_i\mathbf{K}_i\right)/2\sigma^2\right), \tag{6}$$

where $C(n, \sigma, \gamma)$ is a function of $n$, $\sigma$, $\gamma$; $\mathbf{Q}' = \mathbf{Q} + \gamma^{-2}\mathbf{I}_2$, and

$$\mathbf{Q} = \begin{pmatrix} \mathbf{X_T} \cdot \mathbf{X_T} & \mathbf{X_T} \cdot \mathbf{Y_T} \\ \mathbf{Y_T} \cdot \mathbf{X_T} & \mathbf{Y_T} \cdot \mathbf{Y_T} \end{pmatrix}, \mathbf{QK} = \begin{pmatrix} \mathbf{X_T} \cdot \mathbf{X_S} & \mathbf{Y_T} \cdot \mathbf{X_S} \\ \mathbf{X_T} \cdot \mathbf{Y_S} & \mathbf{Y_T} \cdot \mathbf{Y_S} \end{pmatrix} + \gamma^{-2}\mathbf{I}_2. \tag{7}$$

The free parameters are $\gamma$ and the number of 2D rotated copies for each $\mathbf{T}_i$ (since a 2D affine transformation implicitly includes 2D rotations, and since a specific prior probability distribution $\mathbf{N}(\mathbf{X}_0, \gamma\mathbf{I})$ is assumed, both free parameters should be explored together to search for the optimal results).

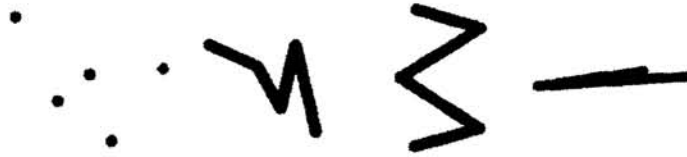

Figure 1: Stimulus classes with increasing structural regularity: Balls, Irregular, Symmetric, and V-Shaped. There were three objects in each class in the experiment.

## 2.5   The human observers

Three naive subjects were tested with four classes of objects: Balls, Irregular, Symmetric, and V-Shaped (Fig. 1). There were three objects in each class. For each object, 11 template views were learned by rotating the object 60°/step, around the X- and Y-axis, respectively. The 2D images were generated by orthographic projection, and viewed monocularly. The viewing distance was 1.5 m. During the test, the standard deviation of the Gaussian noise added to the target object was $\sigma_t = 0.254$ cm. No feedback was provided.

Because the image information available to the humans was more than what was available to the models (shading and occlusion in addition to the $(x, y)$ positions of the vertices), both learned and novel views were tested in a randomly interleaved fashion. Therefore, the strategy that humans used in the task for the learned and novel views should be the same. The number of self-occlusions, which in principle provided relative depth information, was counted and was about equal in both learned and novel view conditions. The shading information was also likely to be equal for the learned and novel views. Therefore, this additional information was about equal for the learned and novel views, and should not affect the comparison of the performance (humans relative to a model) between learned and novel views. We predict that if the humans used a 2D affine strategy, then their performance relative to the 2D affine ideal observer should not be higher for the novel views than for the learned views. One reason to use the four classes of objects with increasing structural regularity is that structural regularity is a 3D property (e.g., 3D Symmetric vs. Irregular), which the 2D models cannot capture. The exception is the planar V-Shaped objects, for which the 2D affine models completely capture 3D rotations, and are therefore the "correct" models. The V-Shaped objects were used in the 2D affine case as a benchmark. If human performance increases with increasing structural regularity of the objects, this would lend support to the hypothesis that humans have used 3D information in the task.

## 2.6   Measuring performance

A stair-case procedure [7] was used to track the observers' performance at 75% correct level for the learned and novel views, respectively. There were 120 trials for the humans, and 2000 trials for each of the models. For the GRBF model, the standard deviation of the Gaussian function was also sampled to search for the best result for the novel views for each of the 12 objects, and the result for the learned views was obtained accordingly. This resulted in a conservative test of the hypothesis of a GRBF model for human vision for the following reasons: (1) Since no feedback was provided in the human experiment and the learned and novel views were randomly intermixed, it is not straightforward for the model to find the best standard deviation for the novel views, particularly because the best standard deviation for the novel views was not the same as that for the learned

ones. The performance for the novel views is therefore the upper limit of the model's performance. (2) The subjects' performance relative to the model will be defined as statistical efficiency (see below). The above method will yield the lowest possible efficiency for the novel views, and a higher efficiency for the learned views, since the best standard deviation for the novel views is different from that for the learned views. Because our hypothesis depends on a higher statistical efficiency for the novel views than for the learned views, this method will make such a putative difference even smaller. Likewise, for the 2D affine ideal, the number of 2D rotated copies of each template $\mathbf{T}_i$ and the value $\gamma$ were both extensively sampled, and the best performance for the novel views was selected accordingly. The result for the learned views corresponding to the same parameters was selected. This choice also makes it a conservative hypothesis test.

## 3   Results

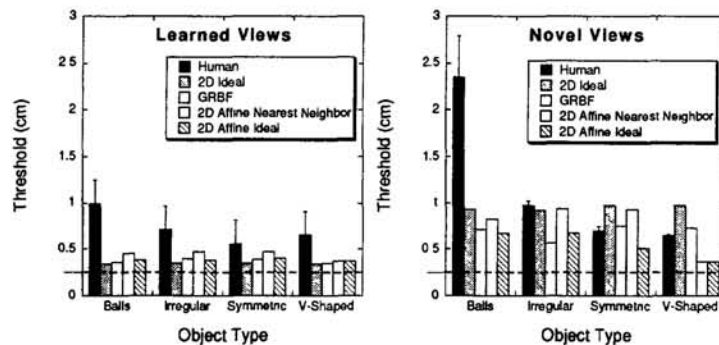

Figure 2: The threshold standard deviation of the Gaussian noise, added to the distractor in the test pair, that keeps an observer's performance at the 75% correct level, for the learned and novel views, respectively. The dotted line is the standard deviation of the Gaussian noise added to the target in the test pair.

Fig. 2 shows the threshold performance. We use statistical efficiency $\mathcal{E}$ to compare human to model performance. $\mathcal{E}$ is defined as the information used by humans relative to the ideal observer [3] : $\mathcal{E} = \left(d'_{human}/d'_{ideal}\right)^2$, where $d'$ is the discrimination index. We have shown in [4] that, in our task, $\mathcal{E} = \left(\left(\sigma^{ideal}_{distractor}\right)^2 - \left(\sigma_{target}\right)^2\right) / \left(\left(\sigma^{human}_{distractor}\right)^2 - \left(\sigma_{target}\right)^2\right)$, where $\sigma$ is the threshold. Fig. 3 shows the statistical efficiency of the human observers relative to each of the four models.

We note in Fig. 3 that the efficiency for the novel views is higher than those for the learned views (several of them even exceeded 100%), except for the planar V-Shaped objects. We are particularly interested in the Irregular and Symmetric objects in the 2D affine ideal case, in which the pairwise comparison between the learned and novel views across the six objects and three observers yielded a significant difference (binomial, $p < 0.05$). This suggests that the 2D affine ideal observer cannot account for the human performance, because if the humans used a 2D affine template matching strategy, their relative performance for the novel views cannot be better than for the learned views. We suggest therefore that 3D information was used by the human observers (e.g., 3D symmetry). This is supported in addition by the increasing efficiencies as the structural regularity increased from the Balls, Irregular, to Symmetric objects (except for the V-Shaped objects with 2D affine models).

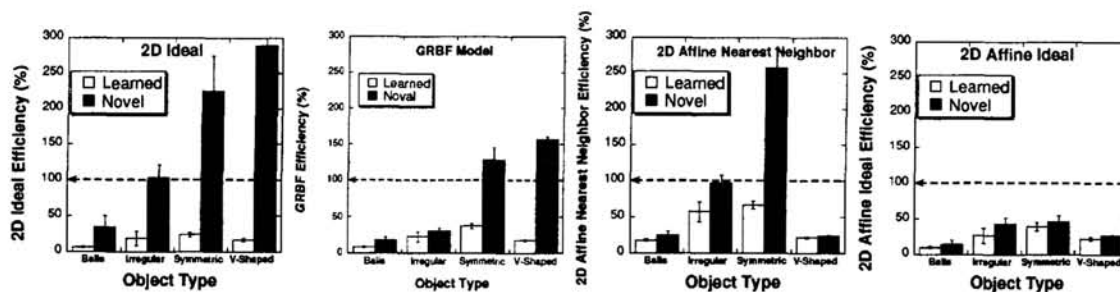

Figure 3: Statistical efficiencies of human observers relative to the 2D ideal observer, the GRBF model, the 2D affine nearest neighbor model, and the 2D affine ideal observer.

## 4 Conclusions

Computational models of visual cognition are subject to information theoretic as well as implementational constraints. When a model's performance mimics that of human observers, it is difficult to interpret which aspects of the model characterize the human visual system. For example, human object recognition could be simulated by both a GRBF model and a model with partial 3D information of the object. The approach we advocate here is that, instead of trying to mimic human performance by a computational model, one designs an implementation-free model for a specific recognition task that yields the best possible performance under explicitly specified computational constraints. This model provides a well-defined benchmark for performance, and if human observers outperform it, we can conclude firmly that the humans must have used better computational strategies than the model. We showed that models of independent 2D templates with 2D linear operations cannot account for human performance. This suggests that our human observers may have used the templates to reconstruct a representation of the object with some (possibly crude) 3D structural information.

## References

[1] Biederman I and Gerhardstein P C. Viewpoint dependent mechanisms in visual object recognition: a critical analysis. J. Exp. Psych.: HPP, 21:1506–1514, 1995.

[2] Bülthoff H H and Edelman S. Psychophysical support for a 2D view interpolation theory of object recognition. Proc. Natl. Acad. Sci., 89:60–64, 1992.

[3] Fisher R A. Statistical Methods for Research Workers. Oliver and Boyd, Edinburgh, 1925.

[4] Liu Z, Knill D C, and Kersten D. Object classification for human and ideal observers. Vision Research, 35:549–568, 1995.

[5] Poggio T and Edelman S. A network that learns to recognize three-dimensional objects. Nature, 343:263–266, 1990.

[6] Tarr M J and Bülthoff H H. Is human object recognition better described by geon-structural-descriptions or by multiple-views? J. Exp. Psych.: HPP, 21:1494–1505, 1995.

[7] Watson A B and Pelli D G. QUEST: A Bayesian adaptive psychometric method. Perception and Psychophysics, 33:113–120, 1983.

[8] Werman M and Weinshall D. Similarity and affine invariant distances between 2D point sets. IEEE PAMI, 17:810–814, 1995.
